# A mechanistic model of early sensory processing based on subtracting sparse representations

**Shaul Druckmann***          **Tao Hu***          **Dmitri B. Chklovskii**

\* - Equal contribution
Janelia Farm Research Campus
{druckmanns, hut, mitya}@janelia.hhmi.org

## Abstract

Early stages of sensory systems face the challenge of compressing information from numerous receptors onto a much smaller number of projection neurons, a so called communication bottleneck. To make more efficient use of limited bandwidth, compression may be achieved using predictive coding, whereby predictable, or redundant, components of the stimulus are removed. In the case of the retina, Srinivasan et al. (1982) suggested that feedforward inhibitory connections subtracting a linear prediction generated from nearby receptors implement such compression, resulting in biphasic center-surround receptive fields. However, feedback inhibitory circuits are common in early sensory circuits and furthermore their dynamics may be nonlinear. Can such circuits implement predictive coding as well? Here, solving the transient dynamics of nonlinear reciprocal feedback circuits through analogy to a signal-processing algorithm called linearized Bregman iteration we show that nonlinear predictive coding can be implemented in an inhibitory feedback circuit. In response to a step stimulus, interneuron activity in time constructs progressively less sparse but more accurate representations of the stimulus, a temporally evolving prediction. This analysis provides a powerful theoretical framework to interpret and understand the dynamics of early sensory processing in a variety of physiological experiments and yields novel predictions regarding the relation between activity and stimulus statistics.

## 1 Introduction

Receptor neurons in early sensory systems are more numerous than the projection neurons that transmit sensory information to higher brain areas, implying that sensory signals must be compressed to pass through a limited bandwidth channel known as "Barlow's bottleneck" [1]. Since natural signals arise from physical objects, which are contiguous in space and time, they are highly spatially and temporally correlated [2-4]. Such signals are ideally suited for predictive coding, a compression strategy borrowed from engineering whereby redundant, or predictable components of the signal are subtracted and only the residual is transmitted [5].

Consider, for example, the processing of natural images in the retina. Instead of transmitting photoreceptor signals, which are highly correlated in space and time, ganglion cells can transmit differences in signal between nearby pixels or consecutive time points. The seminal work of Srinivasan et al. introduced predictive coding to neuroscience, proposing that feedforward inhibition could implement predictive coding by subtracting a prediction for the activity of a given photoreceptor generated from the activity of nearby receptors [6]. Indeed, the well known center surround spatial receptive fields or biphasic temporal receptive fields of ganglion cells [7] may be viewed as evidence of predictive coding because they effectively code such differences [6, 8-10]. Although the Srinivasan et

al. model captured the essence of predictive coding it does not reflect two important biological facts. First, in the retina, and other early sensory systems, inhibition has a significant feedback component [11-13]. Second, interneuron transfer functions are often non-linear [14-16].

Here, we demonstrate that feedback circuits can be viewed as implementing predictive coding. Surprisingly, by taking advantage of recent developments in applied mathematics and signal processing we are able to solve the non-linear recurrent dynamics of such a circuit, for an arbitrary number of sensory channels and interneurons, allowing us to address in detail the circuit dynamics and consequently the temporal and stimulus dependencies. Moreover, introducing non-linear feedback dramatically changes the nature of predictions. Instead of a static relation between stimulus and prediction, we find that the prediction becomes both stimulus and time dependent.

## 2 Model

### 2.1 Dynamics of the linear single-channel feedback circuit

We start by considering predictive coding in feedback circuits, where principal neurons are reciprocally connected with inhibitory interneuron forming a negative feedback loop. Much of the intuition can be developed from linear circuits and we start from this point. Consider a negative feedback circuit composed of a single principal neuron, p, and a single interneuron, n (Fig. 1a). Assuming that both types of neurons are linear first-order elements, their dynamics are given by:

$$C_p^m \frac{dp}{dt} = -g_p^m p(t) + g_p^s(s(t) - wn(t)),$$
$$C_n^m \frac{dn}{dt} = -g_n^m n(t) + g_n^s wp(t) \tag{1}$$

where $g^m$ is the membrane conductance (inverse of membrane resistance), $C^m$ the membrane capacitance, $g^s$ synaptic conductance and the subscript designates the neuron class (principal and interneuron) and $w$ in the second equation is the weight of the synapse from the principal neuron to the interneuron. For simplicity, we assumed that the weight of the synapse from the interneuron to the principal neuron is the same in magnitude but with negative sign, $-w$. Although we do not necessarily expect the brain to fully reconstruct the stimulus on the receiving side, we must still ensure that the transmitted signal is decodable. To guarantee that this is the case, the prediction made by the interneuron must be strictly causal. In other words, there must be a delay between the input to the interneurons, $wp(t)$, and the output of the interneurons, $n(t + \varepsilon)$. Given that feedback requires signals passing through a synapse, such delay is biologically plausible. When discussing analytical solutions below, we assume that $\varepsilon \to 0$ to avoid clutter and do not explicitly indicate the time dependence of the vectors $p, s$, and $n$. By rearranging the terms in Eq. 1 we obtain:

$$\tau_p \frac{dp}{dt} = -p + g_p^s/g_p^m \, (s - wn),$$
$$\tau_n \frac{dn}{dt} = -n + g_n^s/g_n^m \, wp \tag{2}$$

where $\tau=RC$ is the membrane time constant. Since principal neurons should be able to transmit fast changes in the stimuli, we assume that the time constant of the principal cells is small compared to that of the interneurons. Therefore, we can assume that the first equation reaches equilibrium instantaneously:

$$p = \alpha(s - wn)$$
$$\tau_n \frac{dn}{dt} = -n + g_n^s/g_n^m \, wp \,, \tag{3}$$

where we defined $\alpha = g_p^s/g_p^m$. As the purpose of interneuron integration will be to construct stimulus representation, the integration time should be on the order of the auto-correlation time in the stimulus. Since here we study the simplified case of the semi-infinite step-stimulus, the time constant of the neuron should approach infinity. We assume this occurs by the interneurons having a very large membrane resistance (or correspondingly a very small conductance) and moderate capacitance. Therefore, the leakage term, $-n$, which is the only term in the second line of Eq. 3 that

doesn't grow with the membrane resistance, can be neglected in the dynamics of interneurons. By this assumption and substituting the first equation into the second, we find:

$$p = \alpha(s - wn)$$
$$\tau_n \frac{d\boldsymbol{n}}{dt} = g_n^s/g_n^m \, \alpha w(s - wn). \tag{4}$$

Defining the effective time constant $\delta = C_n^m R_n^s/\alpha$ we have:

$$p = \alpha(s - wn)$$
$$\delta \frac{d\boldsymbol{n}}{dt} = wp \tag{5}$$

In response to a step stimulus: $s(t) = \theta(t)s$, where $\theta(t)$ is the Heavyside function, the dynamics of equation 5 are straightforward to solve, yielding:

$$n(t) = \frac{s}{w} \theta(t) \left( 1 - \exp\left( -w^2 \frac{\alpha}{\delta} t \right) \right)$$
$$p(t) = \alpha s \, \theta(t) \exp\left( -w^2 \frac{\alpha}{\delta} t \right) \tag{6}$$

The interneuron's activity, *n(t)*, grows with time as it integrates the output of the principal neuron, *p(t)*, Fig. 1a. In turn, the principal neuron's output, *p(t)*, is the difference between the incoming stimulus and the interneuron's activity, *n(t)*, i.e. a residual, which decays with time from the onset of the stimulus. In the limit considered here (infinite interneuron time constant), the interneuron's feedback will approach the incoming stimulus and the residual will decay to zero. To summarize, one can view the interneuron's activity as a series of progressively more accurate predictions of the stimulus. The principal neuron subtracts these predictions and sends the series of residuals to higher brain areas, a more efficient approach than direct transmission (Fig. 1a).

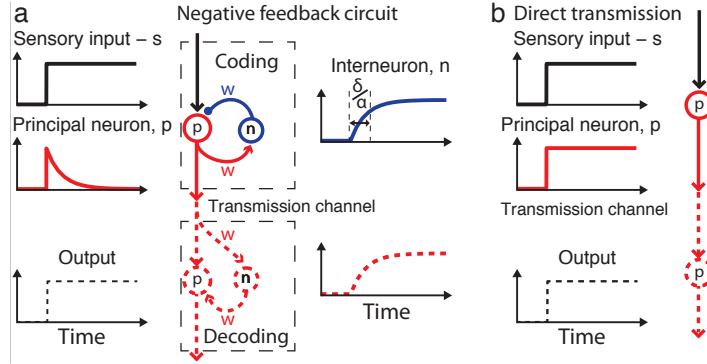

**Figure 1 Schematic view of early processing in a single sensory channel in response to a step stimulus. a.** A predictive coding model consists of a coding circuit, transmission channel and, for theoretical analysis only, a virtual decoding circuit. Coding is performed in a negative feedback circuit containing a principal neuron, *p*, and an inhibitory interneuron, *n*. In response to a step-stimulus (top left) the interneuron charges up with time (top right) till it reaches the value of the stimulus. Principal neuron (middle left) transmits the difference between the interneuron activity and the stimulus, resulting in a transient signal. **b.** Direct transmission.

The transient response to a step stimulus (Fig. 1a left) is consistent with electrophysiological measurements from principal neurons in invertebrate and vertebrate retina [10, 17]. For example, in flies, cells post-synaptic to photoreceptors (the LMCs) have graded potential response consistent with Equation 5. In the vertebrate retina, most recordings are performed on ganglion cells, which read out signals from bipolar cells. In response to a step-stimulus the firing rate of ganglion cells is consistent with Equation 6 [17].

## 2.2 Dynamics of the linear multi-channel feedback circuit

In most sensory systems, stimuli are transmitted along multiple parallel sensory channels, such as mitral cells in the olfactory bulb, or bipolar cells in the retina. Although a circuit could implement predictive coding by replicating the negative feedback loop in each channel, this

solution is likely suboptimal due to the contiguous nature of objects in space, which often results in stimuli correlated across different channels. Therefore, interneurons that combine inputs across channels may generate an accurate prediction more rapidly. The dynamics of a multichannel linear negative feedback circuit are given by:

$$p = s - Wa$$
$$\delta \frac{dn}{dt} = W^T p \tag{7}$$

where boldface lowercase letters are column vectors representing stimulus, $s = (s_1, s_2, s_3, \dots)^T$, activity of principal neurons, $p$, and interneurons, $n$, Fig. 2a. Boldface uppercase letters designate synaptic weight matrices. Synaptic weights from principal neurons to interneurons are $W^T$, and synaptic weights from interneurons to principal neurons are, for simplicity, symmetric but with the negative sign, $-W$. Such symmetry was suggested for olfactory bulb, considering dendro-dendritic synapses [18]. Each column of $W$ contains the weights of synapses from correlated principal neurons to a given interneuron, thus defining that interneuron's feature vector (Fig. 2b).

Linear dynamics of the feedback circuit in response to a multi-dimensional step stimulus can be solved in the standard manner similarly to equation 6:

$$n = (W^T W)^{-1} \left( 1 - \exp\left(-W^T W \frac{\alpha}{\delta} t\right) \right) W^T s$$
$$p = \alpha \left[ 1 - W(W^T W)^{-1} \left( 1 - \exp\left(-W^T W \frac{\alpha}{\delta} t\right) \right) W^T \right] s \tag{8}$$

provided $W^T W$ is invertible. When the matrix $W^T W$ is not full rank, for instance if the number of interneurons exceeds the number of sensory channels, the solution of Equation 7 is given by:

$$n = W^T (WW^T)^{-1} \left( 1 - \exp\left(-WW^T \frac{\alpha}{\delta} t\right) \right) s$$
$$p = \alpha \exp\left(-WW^T \frac{\alpha}{\delta} t\right) s \tag{9}$$

Recapitulating the equations in words, as above one can view the interneurons' activity as a series of progressively more accurate stimulus predictions, $\hat{s} = Wn$. The principal neuron sends the series of residuals of these predictions, $p = s - \hat{s}$, to higher brain areas, and the dynamics result in the transmitted residual decreasing in time [19-22] (Fig. 2c,d).

## 2.3 Dynamics of the non-linear multi-channel feedback circuit

Our solution of the circuit dynamics in the previous sub-section relied on the assumption that neurons act as linear elements, which in view of non-linearities in real neurons, represents a drastic simplification. We now extend this analysis to the non-linear circuit. A typical neural response non-linearity is the existence of a non-zero input threshold below which neurons do not respond. A pair of such on- and off- neurons is described by a threshold function (Fig. 2e) that has a "gap" or "deadzone" around zero activity and is not equivalent to a linear neuron:

$$\text{Thresh}(n) = \begin{cases} n - \lambda, & n > \lambda \\ 0, & |n| \leq \lambda \\ n + \lambda, & n < -\lambda \end{cases} \tag{10}$$

Accordingly, the dynamics are given by:

$$p = s - Wa$$
$$\delta \frac{dn}{dt} = W^T p$$
$$a = \text{Thresh}_\lambda(n) \tag{11}$$

The central contribution of this paper is an analysis of predictive coding in a feedback circuit with threshold-linear interneurons inspired by the equivalence of the network dynamics to a signal-processing algorithm called linearized Bregman iteration [23, 24]. Before showing the equivalence, we first describe linearized Bregman iteration. This algorithm constructs a faithful representation of an input as a linear sum over dictionary elements while minimizing the $L_1$-$L_2$ norm of the representation [25]. Formally, the problem is defined as follows:

$$\text{for } J(\boldsymbol{a}) \equiv \mu||\boldsymbol{a}||_1 + \frac{1}{2\delta}||\boldsymbol{a}||_2^2, \min_{\boldsymbol{a}}\{J(\boldsymbol{a})\} \, s.t. \, \boldsymbol{Wa} = \boldsymbol{s}. \tag{12}$$

Remarkably, this high-dimensional non-linear optimization problem can be solved by a simple iterative scheme (see Appendix):

$$\begin{aligned}\boldsymbol{n}^{k+1} &= \boldsymbol{n}^k + \delta \boldsymbol{W}^T(\boldsymbol{s} - \boldsymbol{W}\boldsymbol{a}^k) \\ \boldsymbol{a}^{k+1} &= \text{Thresh}_\lambda(\boldsymbol{n}^{k+1})\end{aligned}, \tag{13}$$

combining a linear step, which looks like gradient descent on the representation error, and a component-wise threshold-linear step.

Eq. 11, the network dynamics, is the continuous version of linearized Bregman iteration, Eq. 13. Intuitively speaking, the dynamics of the network play the role of the iterations in the algorithm. Having identified this equivalence, we are able to both solve and interpret the transient non-linear dynamics (see supplementary materials for further details). The analytical solution allows us a deeper understanding, for instance of the convergence of the algorithm. We note that if the interneuron feature vectors span the stimulus space the steady-state activity will be zero for any stimulus and thus non-informative. Therefore, solving the transient dynamics, as opposed to just the steady-state activity [18, 19, 21, 26], was particularly crucial in this case.

Next, we describe in words the mathematical expressions for the response of the feedback circuit to a step-stimulus (see Supplement for dynamics equations), Fig. 2f-g. Unlike in the linear circuit, interneurons do not inhibit principal neurons until their internal activity crosses threshold, Fig. 2f. Therefore, their internal activity initially grows with a rate proportional to the projection of the sensory stimulus on their feature vectors, $\boldsymbol{W}^T\boldsymbol{s}$. With time, interneurons cross threshold and contribute to the stimulus representation, thereby constructing a more accurate representation of the stimulus, Fig. 2f,g. The first interneuron to cross threshold is the one for which the projection of the sensory stimulus on its feature vector, $\boldsymbol{W}^T\boldsymbol{s}$ is highest. As its contribution is subtracted from the activity of the principal neurons, the driving force on other interneurons $\boldsymbol{W}^T(\boldsymbol{s} - \boldsymbol{W}\boldsymbol{a})$ changes. Therefore, the order by which interneurons cross threshold depends also on the correlation between the feature vectors, Fig. 2b,f.

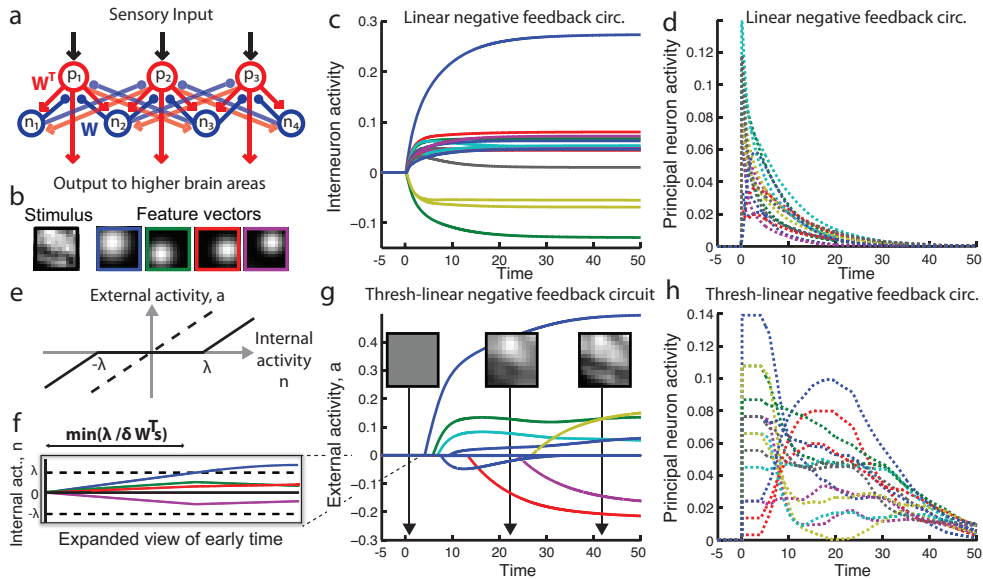

**Figure 2. Predictive coding in a feedback circuit in response to a step stimulus at time zero. a.** Circuit diagram for feedback circuit. **b.** Stimulus (grayscale in black box left) and a subset of interneuron's feature vector (grayscale in boxes). **c-d.** Response of linear feedback circuit to a step stimulus at time zero in interneurons (**c**) and principal neurons (**d**). **e.** Threshold-linear transfer function relating internal, $n$, and external, $a$, activity of interneurons. Dashed line shows diagonal. Firing rates cannot be negative and therefore the threshold-linear function

should be thought as combining a pair of on and off-cells. **f-h**. Response of interneurons (**f-g**) and principal neurons to a step stimulus at time zero. **f.** Expanded view of internal activity of the interneurons (only some are shown, see grayscale in boxes color coded to match b) at early times. **g.** External activity of a larger subset of interneurons over a longer time period. Grayscale boxes show the stimulus represented by the interneuron layer at various times marked by arrows. **h.** Principal neuron activity as a function of time. As interneurons cross threshold they more closely represent the stimulus and cancel out more of the principal cell activity. Eventually, the interneuron representation (right box in **g**) is nearly identical to the stimulus and the principal neurons' activity drops almost to zero.

Collectively the representation progresses from sparse to dense, but individual interneurons may first be active then become silent. Eventually interneurons will accurately represent the input with their activity, $s = Wa$, and will fully subtract it from the principal cells' activity, resulting in no further excitation to the interneurons, Fig. 2g,h.

However, this description leads to an immediate puzzle. Namely, the algorithm builds a representation of the stimulus by the activity of interneurons. Yet, interneurons are local circuit elements whose activity is not transmitted outside the circuit. Why would a representation be built if it is available only locally within the neural circuit? The answer to this conundrum is found by considering the notion of predictive coding in early sensory circuits presented in the introduction. The interneurons serve as the predictor and the principal neurons transmit a prediction residual.

As expected by the framework of predictive coding, at each point in time, the circuit subtracts the prediction, $\hat{s} = Wa$, which was constructed in the interneurons from previous incoming sensory signals, from the current sensory stimulus and the principal neurons transmit the residual, $p = s - \hat{s}$, to higher brain areas. We note that initially the interneurons are silent and the principal neurons transmit the stimulus directly. If there were no bandwidth limitation, the stimulus could be decoded just from this initial transmission. However, the bandwidth limitation results in coarse, or noisy, principal neuron transmission, an issue we will return to later.

## 3 Results

In neuroscience, the predictive coding strategy was originally suggested to allow efficient transmission through a limited bandwidth channel (Srinivasan et al., 1982). Our main result is the solution of the transient dynamics given in the section above. Understanding circuit dynamics in the predictive coding framework allows us to make a prediction regarding the length of transient activity for different types of stimuli. Recall that the time from stimulus onset to cancellation of the stimulus depends on the rate of the interneurons' activation, which in turn is proportional to the projection of the stimulus on the interneurons' feature vectors. Presumably, interneuron feature vectors are adapted to the most common stimuli, e.g. natural images in the case of the retina, therefore this type of stimulus should be relatively quickly cancelled out. In contrast, non-natural stimuli, like white noise patterns, will be less well captured by interneuron receptive fields and their activation will occur after a longer delay. Accordingly, it will take longer to cancel out non-natural stimuli, leading to longer principal neuron transients.

Below, we show that the feedback circuit with threshold-linear neurons is indeed more efficient than the existing alternatives. We first consider a scenario in which effective bandwidth limitation is imposed through addition of noise. Secondly, we consider a more biologically relevant model, where transmission bandwidth is set by the discreteness of Poisson neural activity.

We find that threshold linear interneurons achieve more accurate predictions when faced with stimulus corrupted with i.i.d Gaussian noise. The intuition behind this result is that of sparse denoising [23]. Namely, if the signal can be expressed as a sparse sum of strong activation of dictionary elements, whereas the noise requires a large number of weakly activated elements, then thresholding the elements will suppress the noise more than the signal, yielding denoising. We note that this fact alone does not in itself argue for the biological plausibility of this network, but threshold-linear dynamics are a common approximation in neural networks.

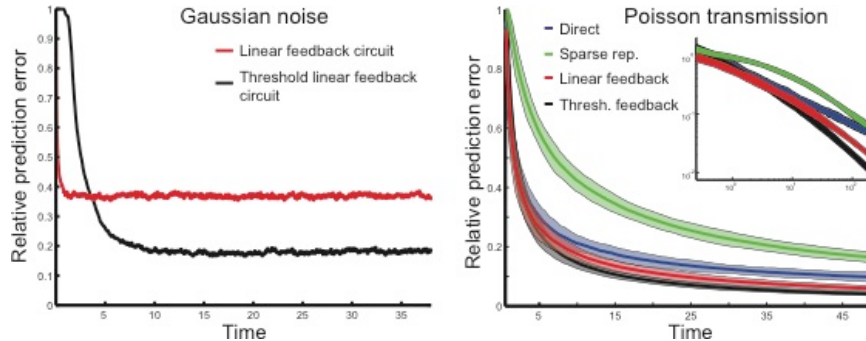

**Figure 3. Predictions by negative feedback circuit. Left:** Relative prediction error ($\|\boldsymbol{s} - \hat{\boldsymbol{s}}\|_2 / \|\boldsymbol{s}\|_2$), where $\hat{\boldsymbol{s}} = \boldsymbol{W}\boldsymbol{n}$, as a function of time for a stimulus consisting of an image patch corrupted by i.i.d Gaussian noise at every time point. **Right:** An image is sent through principal neurons that transmit Poisson. The reconstruction error as a function of time following the presentation of stimulus is shown for the full non-linear negative feedback circuit (black), for a linear negative feedback circuit (red), for a direct transmission circuit (blue), and for a circuit where the sparse approximation itself is transmitted instead of the residual (green). Time on the x-axis is measured in units of the time length in which a single noisy transmission occurs. Inset shows log-log plot.

In addition to considering transmission of stimuli corrupted by Gaussian noise, we also studied a different model where bandwidth limitation is set by the discreteness of spiking, modeled by a Poisson process. Although the discreteness of transmission can be overcome by averaging over time, this comes at the cost of longer perceptual delays, or lower transmission rates, as longer integration takes place. Therefore, we characterize transmission efficiency by reconstruction error as a function of time, Fig. 3. We find that, for Poisson transmission, predictive coding provides more accurate stimulus reconstruction than direct transmission for all times but the brief interval until the first interneuron has crossed threshold (Fig. 3).

## 4 Discussion

By solving the dynamics of the negative feedback circuit through equivalence to linearized Bregman iteration we have shown that the development of activity in a simplified early sensory circuit can be viewed as implementing an efficient, non-linear, intrinsically parallel algorithm for predictive coding. Our study maps the steps of the algorithm onto specific neuronal substrates, providing a solid theoretical framework for understanding physiological experiments on early sensory processing as well as experimentally testing predictive coding ideas on a finer, more quantitative level.

Recently, sparse representations were studied in a single-layer circuit with lateral inhibitory connections proposed as a model of a different brain area, namely primary cortical areas. The circuit constructs the stimulus representation in the projection neurons themselves and directly transmits it downstream [27, 28]. We believe it does not model early sensory systems as well as the negative feedback circuit for a number of reasons. First, anatomical data is more consistent with the reciprocally connected interneuron layer than lateral connections between principal neurons [11, 13]. Second, direct transmission of the representation would result in greater perceptual delays after stimulus onset since no information is transmitted while the representation is being built up in the sub-threshold range. In contrast, in the predictive coding model the projection neurons pass forth (a coarse and possibly noisy version of) the input stimulus from the very beginning. We note that adding a nonlinearity on the principal neurons would result in a delay in transmission in both models. Although there is no biological justification for introducing a threshold to interneurons only, the availability of an analytically solvable model justifies this abstraction. Dynamics of a circuit with threshold on principal neurons will be explored elsewhere.

From a computational point of view there are three main advantages to overcompleteness in the negative feedback circuit. First, the delay until subtraction of

prediction, which occurs when the first interneuron crosses threshold, will be briefer as the number of feature vectors grows since the maximal projection of the stimulus on the interneurons' feature vectors will be higher. Second, the larger the number of feature vectors the fewer the number of interneurons with supra-threshold activity, which may be energetically more efficient. Third, if stimuli come from different statistical ensembles, it could be advantageous to have feature vectors tailored to the different stimulus ensembles, which may result in more feature vectors, i.e., interneurons than principle neurons.

Our study considered responses to step-like stimuli. If the sensory environment changes on slow time scales, a series of step-like responses may be taken as an approximation to the true signal. Naturally, the extension of our framework to fully time-varying stimuli is an important research direction.

**Acknowledgements**
We thank S. Baccus, A. Genkin, V. Goyal, A. Koulakov, M. Meister, G. Murphy, D. Rinberg, and R. Wilson for valuable discussions and their input.

**Appendix: Derivation of linearized Bregman iteration**
Here, inspired by [22,23], we solve the following basis pursuit-like optimization problem:

For $J(\boldsymbol{a}) \equiv \mu||\boldsymbol{a}||_1 + \frac{1}{2\delta}||\boldsymbol{a}||_2^2, \min_a\{J(\boldsymbol{a})\}\ s.t.\boldsymbol{Wa} = \boldsymbol{s}$. $\qquad$ (A1)

The idea behind linearized Bregman iteration, is to start with $\boldsymbol{a}^0 = 0$ and, at each iteration, to seek to update $\boldsymbol{a}$ so as to minimize the square error plus the distance from the previous value of $\boldsymbol{a}$. Thus, we perform the following update:

$$\boldsymbol{a}^{k+1} = \text{argmin}_a \left\{ D_J^{\boldsymbol{p}^k}(\boldsymbol{a}, \boldsymbol{a}^k) + \frac{1}{2}||\boldsymbol{s} - \boldsymbol{Wa}||^2 \right\} \qquad (A2)$$

where we used a notation $D_J^{\boldsymbol{p}}(\boldsymbol{a}, \boldsymbol{b})$ for the Bregman divergence [29] between the two points $\boldsymbol{a}$ and $\boldsymbol{b}$ induced by the convex function $J$. The Bregman divergence is an appropriate measure for such problems that can handle the non-differentiable nature of the cost. It is defined by the following expression: $D_J^{\boldsymbol{p}}(\boldsymbol{a}, \boldsymbol{b}) = J(\boldsymbol{a}) - J(\boldsymbol{b}) - \langle \boldsymbol{p}, \boldsymbol{a} - \boldsymbol{b} \rangle$, where $\boldsymbol{p} \in \partial J(\boldsymbol{b})$ is an element of the subgradient of $J$ at the point $\boldsymbol{b}$.

The Bregman divergence for the elastic net cost function $J$ defined in Eq. A1 is:

$$D_J^{\boldsymbol{p}}(\boldsymbol{a}, \boldsymbol{a}^k) = \mu||\boldsymbol{a}||_1 - \mu||\boldsymbol{a}^k||_1 + \frac{1}{2\delta}||\boldsymbol{a}||_2^2 - \frac{1}{2\delta}|\boldsymbol{a}^k||_2^2 - \langle \boldsymbol{p}, \boldsymbol{a} - \boldsymbol{a}^k \rangle, \qquad (A3)$$

where $\boldsymbol{p}$ is a subgradient of $J$ at $\boldsymbol{a}^k$. The condition for the minimum in Eq. A2 is:

$$\partial \left[ \mu||\boldsymbol{a}^{k+1}||_1 + \frac{1}{2\delta}|\boldsymbol{a}^{k+1}||_2^2 \right] \ni \boldsymbol{p}^k + \boldsymbol{W}^T(\boldsymbol{s} - \boldsymbol{Wa}^k), \qquad (A4)$$

where $\partial\,[.]$ designates a subdifferential. Consistency of the iteration scheme requires that the update $\boldsymbol{p}^{k+1}$ be a subgradient of $J$ as well:

$$\partial \left[ \mu||\boldsymbol{a}^{k+1}||_1 + \frac{1}{2\delta}|\boldsymbol{a}^{k+1}||_2^2 \right] \ni \boldsymbol{p}^{k+1}. \qquad (A5)$$

By combining Eqs. A4,A5 we set:

$$\boldsymbol{p}^{k+1} = \boldsymbol{p}^k + \boldsymbol{W}^T(\boldsymbol{s} - \boldsymbol{Wa}^k). \qquad (A6)$$

By substituting Eq. A6 into Eq. A4 and simplifying we get:

$$\boldsymbol{a}^{k+1} = \text{argmin}_u \left\{ \mu||\boldsymbol{a}||_1 + \frac{1}{2\delta}||\boldsymbol{a} - \delta\boldsymbol{p}^{k+1}||^2 \right\}, \qquad (A7)$$

which has the explicit solution:

$$\boldsymbol{a}^{k+1} = \text{Thresh}_{\delta\mu}(\delta\boldsymbol{p}^{k+1}) \qquad (A8)$$

By defining $\boldsymbol{n}^k = \delta\boldsymbol{p}^k$ and expressing it in Eqs. A6,A8 with substitution $\lambda = \mu\delta$ we get:

$$\boldsymbol{n}^{k+1} = \boldsymbol{n}^k + \delta\boldsymbol{W}^T(\boldsymbol{s} - \boldsymbol{Wa}^k)$$
$$\boldsymbol{a}^{k+1} = \text{Thresh}_\lambda(\boldsymbol{n}^{k+1}) \qquad (A9)$$

Eq. A9 is the linearized Bregman iteration algorithm (main text Eq. 13), thereby showing that the iterative scheme indeed finds a minimum of Eq. A2 at every time point. The sequence convergence proof [23, 24] is beyond the scope of this paper.

**References**

1.  Barlow, H.B. and W.R. Levick, *Threshold setting by the surround of cat retinal ganglion cells.* The Journal of physiology, 1976. **259**(3): p. 737-57.
2.  Dong, D.W. and J.J. Atick, *Statistics of natural time-varying images.* Network: Computation in Neural Systems, 1995. **6**(3): p. 345--358.
3.  Field, D.J., *Relations between the statistics of natural images and the response properties of cortical cells.* Journal of the Optical Society of America. A, Optics and image science, 1987. **4**(12): p. 2379-94.
4.  Ruderman, D.L. and W. Bialek, *Statistics of natural images: Scaling in the woods.* Physical review letters, 1994. **73**(6): p. 814-817.
5.  Elias, P., *Predictive coding.* Information Theory, IRE Transactions on, 1955. **1**(1): p. 16--24.
6.  Srinivasan, M.V., S.B. Laughlin, and A. Dubs, *Predictive coding: a fresh view of inhibition in the retina*, in *Proc R Soc Lond, B, Biol Sci* 1982. p. 427-59.
7.  Victor, J.D., *Temporal aspects of neural coding in the retina and lateral geniculate.* Network-Computation in Neural Systems, 1999. **10**(4): p. R1-R66.
8.  Hosoya, T., S.A. Baccus, and M. Meister, *Dynamic predictive coding by the retina.* Nature, 2005. **436**(7047): p. 71-7.
9.  Huang, Y. and R.P.N. Rao, *Predictive coding.* Wiley Interdisciplinary Reviews: Cognitive Science, 2011. **2**(5): p. 580-593.
10. Laughlin, S., *A simple coding procedure enhances a neuron's information capacity.* Zeitschrift fur Naturforschung. Section C: Biosciences, 1981. **36**(9-10): p. 910-2.
11. Masland, R.H., *The fundamental plan of the retina.* Nature neuroscience, 2001. **4**(9): p. 877-86.
12. Olsen, S.R., V. Bhandawat, and R.I. Wilson, *Divisive normalization in olfactory population codes.* Neuron, 2010. **66**(2): p. 287-99.
13. Shepherd, G.M., et al., *The olfactory granule cell: from classical enigma to central role in olfactory processing.* Brain research reviews, 2007. **55**(2): p. 373-82.
14. Arevian, A.C., V. Kapoor, and N.N. Urban, *Activity-dependent gating of lateral inhibition in the mouse olfactory bulb.* Nature neuroscience, 2008. **11**(1): p. 80-7.
15. Baccus, S.A., *Timing and computation in inner retinal circuitry.* Annu Rev Physiol, 2007. **69**: p. 271-90.
16. Rieke, F. and G. Schwartz, *Nonlinear spatial encoding by retinal ganglion cells: when 1+1 not equal 2.* Journal of General Physiology, 2011. **138**(3): p. 283-290.
17. Shapley, R.M. and J.D. Victor, *The effect of contrast on the transfer properties of cat retinal ganglion cells.* The Journal of physiology, 1978. **285**: p. 275-98.
18. Koulakov, A.A. and D. Rinberg, *Sparse Incomplete Representations: A Potential Role of Olfactory Granule Cells.* Neuron, 2011. **72**(1): p. 124-136.
19. Lee, D.D. and H.S. Seung, *Unsupervised learning by convex and conic coding.* Advances in Neural Information Processing Systems, 1997: p. 515--521.
20. Lochmann, T. and S. Deneve, *Neural processing as causal inference.* Curr Opin Neurobiol, 2011.
21. Olshausen, B.A. and D.J. Field, *Sparse coding with an overcomplete basis set: a strategy employed by V1?* Vision research, 1997. **37**(23): p. 3311-25.
22. Rao, R.P.N. and D.H. Ballard, *Predictive coding in the visual cortex: a functional interpretation of some extra-classical receptive-field effects.* nature neuroscience, 1999. **2**: p. 79--87.
23. Osher, S., et al., *Fast linearized Bregman iteration for compressive sensing and sparse denoising.* Communications in Mathematical Sciences, 2009.
24. Yin, W., et al., *Bregman iterative algorithms for l1-minimization with applications to compressed sensing.* SIAM Journal on Imaging Sciences, 2008. **1**(1): p. 143--168.
25. Zou, H. and T. Hastie, *Regularization and variable selection via the elastic net.* Journal of the Royal Statistical Society: Series B (Statistical Methodology), 2005. **67**(2): p. 301--320.
26. Dayan, P., *Recurrent sampling models for the Helmholtz machine.* Neural computation, 1999. **11**(3): p. 653-78.
27. Rehn, M. and F.T. Sommer, *A network that uses few active neurones to code visual input predicts the diverse shapes of cortical receptive fields.* Journal of computational neuroscience, 2007. **22**(2): p. 135-46.
28. Rozell, C.J., et al., *Sparse coding via thresholding and local competition in neural circuits.* Neural computation, 2008. **20**(10): p. 2526-63.
29. Bregman, L.M., *The relaxation method of finding the common point of convex sets and its application to the solution of problems in convex programming* 1.* USSR computational mathematics and mathematical physics, 1967. **7**(3): p. 200--217.
